# Beating a Defender in Robotic Soccer: Memory-Based Learning of a Continuous Function

**Peter Stone**
Department of Computer Science
Carnegie Mellon University
Pittsburgh, PA 15213

**Manuela Veloso**
Department of Computer Science
Carnegie Mellon University
Pittsburgh, PA 15213

## Abstract

Learning how to adjust to an opponent's position is critical to the success of having intelligent agents collaborating towards the achievement of specific tasks in unfriendly environments. This paper describes our work on a Memory-based technique for to choose an action based on a continuous-valued state attribute indicating the position of an opponent. We investigate the question of how an agent performs in nondeterministic variations of the training situations. Our experiments indicate that when the random variations fall within some bound of the initial training, the agent performs better with some initial training rather than from a tabula-rasa.

## 1  Introduction

One of the ultimate goals subjacent to the development of intelligent agents is to have multiple agents collaborating in the achievement of tasks in the presence of hostile opponents. Our research works towards this broad goal from a Machine Learning perspective. We are particularly interested in investigating how an intelligent agent can choose an action in an adversarial environment. We assume that the agent has a specific goal to achieve. We conduct this investigation in a framework where teams of agents compete in a game of robotic soccer. The real system of model cars remotely controlled from off-board computers is under development. Our research is currently conducted in a simulator of the physical system.

Both the simulator and the real-world system are based closely on systems designed by the Laboratory for Computational Intelligence at the University of British Columbia [Sahota *et al.*, 1995, Sahota, 1993]. The simulator facilitates the control of any number of cars and a ball within a designated playing area. Care has been taken to ensure that the simulator models real-world responses (friction, conserva-

tion of momentum, etc.) as closely as possible. Figure 1(a) shows the simulator graphics.

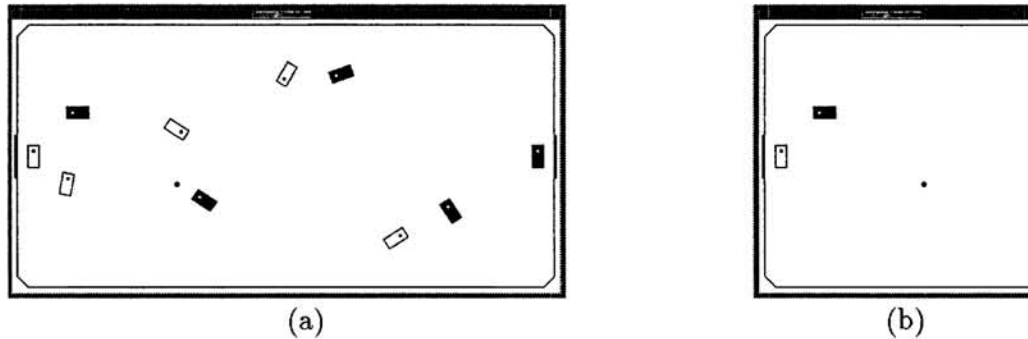

(a)                                              (b)

Figure 1: (a) the graphic view of our simulator. (b) The initial position for all of the experiments in this paper. The teammate (black) remains stationary, the defender (white) moves in a small circle at different speeds, and the ball can move either directly towards the goal or towards the teammate. The position of the ball represents the position of the learning agent.

We focus on the question of learning to choose among actions in the presence of an adversary. This paper describes our work on applying memory-based supervised learning to acquire strategy knowledge that enables an agent to decide how to achieve a goal. For other work in the same domain, please see [Stone and Veloso, 1995b]. For an extended discussion of other work on incremental and memory-based learning [Aha and Salzberg, 1994, Kanazawa, 1994, Kuh *et al.*, 1991, Moore, 1991, Salganicoff, 1993, Schlimmer and Granger, 1986, Sutton and Whitehead, 1993, Wettschereck and Dietterich, 1994, Winstead and Christiansen, 1994], particularly as it relates to this paper, please see [Stone and Veloso, 1995a].

The input to our learning task includes a continuous-valued range of the position of the adversary. This raises the question of how to discretize the space of values into a set of learned features. Due to the cost of learning and reusing a large set of specialized instances, we notice a clear advantage to having an appropriate degree of generalization. For more details please see [Stone and Veloso, 1995a].

Here, we address the issue of the effect of differences between past episodes and the current situation. We performed extensive experiments, training the system under particular conditions and then testing it (with learning continuing incrementally) in nondeterministic variations of the training situation. Our results show that when the random variations fall within some bound of the initial training, the agent performs better with some initial training rather than from a tabula-rasa. This intuitive fact is interestingly well- supported by our empirical results.

## 2   Learning Method

The learning method we develop here applies to an agent trying to learn a function with a continuous domain. We situate the method in the game of robotic soccer.

We begin each trial by placing a ball and a stationary car acting as the "teammate" in specific places on the field. Then we place another car, the "defender," in front of the goal. The defender moves in a small circle in front of the goal at some speed and begins at some random point along this circle. The learning agent must take one of two possible actions: *shoot* straight towards the goal, or *pass* to the teammate so

that the ball will rebound towards the goal. A snapshot of the experimental setup is shown graphically in Figure 1(b).

The task is essentially to learn two functions, each with one continuous input variable, namely the defender's position. Based on this position, which can be represented unambiguously as the angle at which the defender is facing, $\phi$, the agent tries to learn the probability of scoring when shooting, $P_s^*(\phi)$, and the probability of scoring when passing, $P_p^*(\phi)$.[1] If these functions were learned completely, which would only be possible if the defender's motion were deterministic, then both functions would be binary partitions: $P_s^*, P_p^* : [0.0, 360.0) \mapsto \{-1, 1\}$.[2] That is, the agent would know without doubt for any given $\phi$ whether a shot, a pass, both, or neither would achieve its goal. However, since the agent cannot have had experience for every possible $\phi$, and since the defender may not move at the same speed each time, the learned functions must be approximations: $P_s, P_p : [0.0, 360.0) \mapsto [-1.0, 1.0]$.

In order to enable the agent to learn approximations to the functions $P_s^*$ and $P_p^*$, we gave it a memory in which it could store its experiences and from which it could retrieve its current approximations $P_s(\phi)$ and $P_p(\phi)$. We explored and developed appropriate methods of storing to and retrieving from memory and an algorithm for deciding what action to take based on the retrieved values.

## 2.1 Memory Model

Storing every individual experience in memory would be inefficient both in terms of amount of memory required and in terms of generalization time. Therefore, we store $P_s$ and $P_p$ only at discrete, evenly-spaced values of $\phi$. That is, for a memory of size $M$ (with $M$ dividing evenly into 360 for simplicity), we keep values of $P_p(\theta)$ and $P_s(\theta)$ for $\theta \in \{360n/M \mid 0 \leq n < M\}$. We store memory as an array "**Mem**" of size $M$ such that **Mem**$[n]$ has values for both $P_p(360n/M)$ and $P_s(360n/M)$. Using a fixed memory size precludes using memory-based techniques such as K-Nearest-Neighbors (kNN) and kernel regression which require that every experience be stored, choosing the most relevant only at decision time. Most of our experiments were conducted with memories of size 360 (low generalization) or of size 18 (high generalization), i.e. $M = 18$ or $M = 360$. The memory size had a large effect on the rate of learning [Stone and Veloso, 1995a].

### 2.1.1 Storing to Memory

With $M$ discrete memory storage slots, the problem then arises as to how a specific training example should be generalized. Training examples are represented here as $E_{\phi,a,r}$, consisting of an angle $\phi$, an action $a$, and a result $r$ where $\phi$ is the initial position of the defender, $a$ is "s" or "p" for "shoot" or "pass," and $r$ is "1" or "-1" for "goal" or " miss" respectively. For instance, $E_{72.345,p,1}$ represents a pass resulting in a goal for which the defender started at position 72.345° on its circle.

Each experience with $\theta - 360/2M \leq \phi < \theta + 360/2M$ affects **Mem**$[\theta]$ in proportion to the distance $|\theta - \phi|$. In particular, **Mem**$[\theta]$ keeps running sums of the magnitudes of scaled results, **Mem**$[\theta]$.*total-a-results*, and of scaled positive results, **Mem**$[\theta]$.*positive-a-results*, affecting $P_a(\theta)$, where "a" stands for "s" or "p" as before. Then at any given time, $P_a(\theta) = -1 + 2 * \frac{positive-a-results}{total-a-results}$. The "-1" is for

the lower bound of our probability range, and the "2∗" is to scale the result to this range. Call this our *adaptive* memory storage technique:

> Adaptive Memory Storage of $E_{\phi,a,r}$ in **Mem**[$\theta$]
> - $r' = r * (1 - \frac{|\phi - \theta|}{360/M})$.
> - **Mem**[$\theta$].*total-a-results* += $r'$.
> - If $r' > 0$ Then **Mem**[$\theta$].*positive-a-results* += $r'$.
> - $P_a(\theta) = -1 + 2 * \frac{positive-a-results}{total-a-results}$.

For example, $E_{110,p,1}$ would set both *total-p-results* and *positive-p-results* for **Mem**[120] (and **Mem**[100]) to 0.5 and consequently $P_p(120)$ (and $P_p(100)$) to 1.0. But then $E_{125,p,-1}$ would increment *total-p-results* for **Mem**[120] by .75, while leaving *positive-p-results* unchanged. Thus $P_p(120)$ becomes $-1 + 2 * \frac{.5}{1.25} = -.2$.

This method of storing to memory is effective both for time-varying concepts and for concepts involving random noise. It is able to deal with conflicting examples within the range of the same memory slot.

Notice that each example influences 2 different memory locations. This memory storage technique is similar to the kNN and kernel regression function approximation techniques which estimate $f(\phi)$ based on $f(\theta)$ possibly scaled by the distance from $\theta$ to $\phi$ for the k nearest values of $\theta$. In our linear continuum of defender position, our memory generalizes training examples to the 2 nearest memory locations.[3]

### 2.1.2 Retrieving from Memory

Since individual training examples affect multiple memory locations, we use a simple technique for retrieving $P_a(\phi)$ from memory when deciding whether to shoot or to pass. We round $\phi$ to the nearest $\theta$ for which **Mem**[$\theta$] is defined, and then take $P_a(\theta)$ as the value of $P_a(\phi)$. Thus, each **Mem**[$\theta$] represents $P_a(\phi)$ for $\theta - 360/2M \le \phi < \theta + 360/2M$. Notice that retrieval is much simpler when using this technique than when using kNN or kernel regression: we look directly to the closest fixed memory position, thus eliminating the indexing and weighting problems involved in finding the k closest training examples and (possibly) scaling their results.

### 2.2 Choosing an Action

The action selection method is designed to make use of memory to select the action most probable to succeed, and to fill memory when no useful memories are available. For example, when the defender is at position $\phi$, the agent begins by retrieving $P_p(\phi)$ and $P_s(\phi)$ as described above. Then, it acts according to the following function:

> If $P_p(\phi) = P_s(\phi)$ (no basis for a decision), shoot or pass randomly.
> else If $P_p(\phi) > 0$ and $P_p(\phi) > P_s(\phi)$, pass.
>   else If $P_s(\phi) > 0$ and $P_s(\phi) > P_p(\phi)$, shoot.
>     else If $P_p(\phi) = 0$, (no previous passes) pass.
>       else If $P_s(\phi) = 0$, (no previous shots) shoot.
>         else ($P_p(\phi), P_s(\phi) < 0$) shoot or pass randomly.

An action is only selected based on the memory values if these values indicate that one action is likely to succeed and that it is better than the other. If, on the other hand, neither value $P_p(\phi)$ nor $P_s(\phi)$ indicate a positive likelihood of success, then an action is chosen randomly. The only exception to this last rule is when one of

the values is zero,[4] suggesting that there has not yet been any training examples for that action at that memory location. In this case, there is a bias towards exploring the untried action in order to fill out memory.

## 3 Experiments and Results

In this section, we present the results of our experiments. We explore our agent's ability to learn time-varying and nondeterministic defender behavior.

While examining the results, keep in mind that even if the agent used the functions $P_s^*$ and $P_p^*$ to decide whether to shoot or to pass, the success rate would be significantly less than 100% (it would differ for different defender speeds): there were many defender starting positions for which neither shooting nor passing led to a goal (see Figure 2). For example, from our experiments with the defender moving

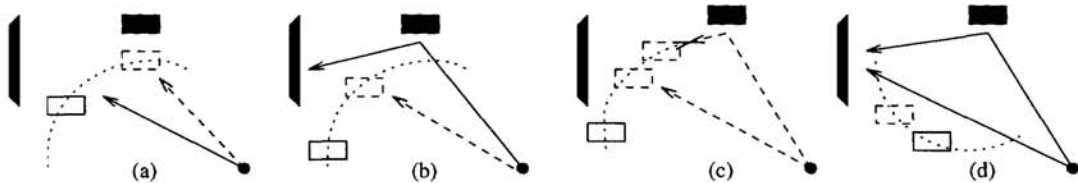

Figure 2: For different defender starting positions (solid rectangle), the agent can score when (a) shooting, (b) passing, (c) neither, or (d) both.

at a constant speed of 50,[5] we found that an agent acting optimally scores 73.6% of the time; an agent acting randomly scores only 41.3% of the time. These values set good reference points for evaluating our learning agent's performance.

### 3.1 Coping with Changing Concepts

Figure 3 demonstrates the effectiveness of adaptive memory when the defender's speed changes. In all of the experiments represented in these graphs, the agent

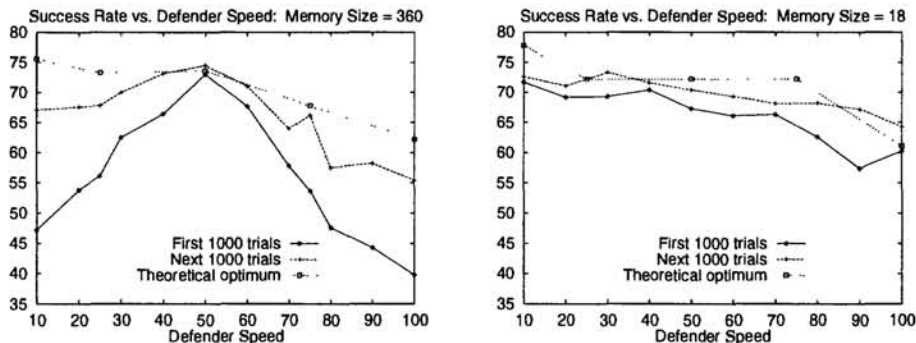

Figure 3: For all trials shown in these graphs, the agent began with a memory trained for a defender moving at constant speed 50.

started with a memory trained by attempting a single pass and a single shot with the defender starting at each position $\theta$ for which $\mathbf{Mem}[\theta]$ is defined and moving in

its circle at speed 50. We tested the agent's performance with the defender moving at various (constant) speeds.

With adaptive memory, the agent is able to unlearn the training that no longer applies and approach optimal behavior: it *re-learns* the new setup. During the first 1000 trials the agent suffers from having practiced in a different situation (especially for the less generalized memory, $M = 360$), but then it is able to approach optimal behavior over the next 1000 trials. Remember that optimal behavior, represented in the graph, leads to roughly a 70% success rate, since at many starting positions, neither passing nor shooting is successful.

From these results we conclude that our adaptive memory can effectively deal with time-varying concepts. It can also perform well when the defender's motion is nondeterministic, as we show next.

## 3.2   Coping with Noise

To model nondeterministic motion by the defender, we set the defender's speed randomly within a range. For each attempt this speed is constant, but it varies from attempt to attempt. Since the agent observes only the defender's initial position, from the point of view of the agent, the defender's motion is nondeterministic.

This set of experiments was designed to test the effectiveness of adaptive memory when the defender's speed was both nondeterministic and different from the speed used to train the existing memory. The memory was initialized in the same way as in Section 3.1 (for defender speed 50). We ran experiments in which the defender's speed varied between 10 and 50. We compared an agent with trained memory against an agent with initially empty memories as shown in Figure 4.

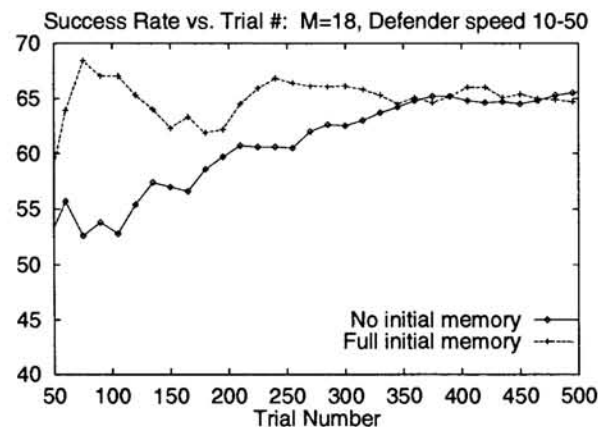

Figure 4: A comparison of the effectiveness of starting with an empty memory versus starting with a memory trained for a constant defender speed (50) different from that used during testing. Success rate is measured as goal percentage thus far.

The agent with full initial memory outperformed the agent with initially empty memory in the short run. The agent learning from scratch did better over time since it did not have any training examples from when the defender was moving at a fixed speed of 50; but at first, the training examples for speed 50 were better than no training examples. Thus, when you would like to be successful immediately upon entering a novel setting, adaptive memory allows training in related situations to be effective without permanently reducing learning capacity.

# 4 Conclusion

Our experiments demonstrated that online, incremental, supervised learning can be effective at learning functions with continuous domains. We found that adaptive memory made it possible to learn both time-varying and nondeterministic concepts. We empirically demonstrated that short-term performance was better when acting with a memory trained on a concept related to but different from the testing concept, than when starting from scratch. This paper reports experimental results on our work towards multiple learning agents, both cooperative and adversarial, in a continuous environment.

Future work on our research agenda includes simultaneous learning of the defender and the controlling agent in an adversarial context. We will also explore learning methods with several agents where teams are guided by planning strategies. In this way we will simultaneously study cooperative and adversarial situations using reactive and deliberative reasoning.

## Acknowledgements

We thank Justin Boyan and the anonymous reviewers for their helpful suggestions. This research is sponsored by the Wright Laboratory, Aeronautical Systems Center, Air Force Materiel Command, USAF, and the Advanced Research Projects Agency (ARPA) under grant number F33615-93-1-1330. The views and conclusions contained in this document are those of the authors and should not be interpreted as necessarily representing the official policies or endorsements, either expressed or implied, of Wright Laboratory or the U. S. Government.

## Footnotes

[1] As per convention, $P^*$ represents the target (optimal) function.

[2] Although we think of $P_s^*$ and $P_p^*$ as functions from angles to probabilities, we will use -1 rather than 0 as the lower bound of the range. This representation simplifies many of our illustrative calculations.

[3] For particularly large values of $M$ it is useful to generalize training examples to more memory locations, particularly at the early stages of learning. However for the values of $M$ considered in this paper, we always generalize to the 2 nearest memory locations.

[4]Recall that a memory value of 0 is equivalent to a probability of .5, representing no reason to believe that the action will succeed or fail.

[5]In the simulator, "50" represents 50 cm/s. Subsequently, we omit the units.

## References

[Aha and Salzberg, 1994] David W. Aha and Steven L. Salzberg. Learning to catch: Applying nearest neighbor algorithms to dynamic control tasks. In P. Cheeseman and R. W. Oldford, editors, *Selecting Models from Data: Artificial Intelligence and Statistics IV*. Springer-Verlag, New York, NY, 1994.

[Kanazawa, 1994] Keiji Kanazawa. Sensible decisions: Toward a theory of decision-theoretic information invariants. In *Proceedings of the Twelfth National Conference on Artificial Intelligence*, pages 973–978, 1994.

[Kuh et al., 1991] A. Kuh, T. Petsche, and R.L. Rivest. Learning time-varying concepts. In *Advances in Neural Information Processing Systems 3*, pages 183–189. Morgan Kaufman, December 1991.

[Moore, 1991] A.W. Moore. Fast, robust adaptive control by learning only forward models. In *Advances in Neural Information Processing Systems 3*. Morgan Kaufman, December 1991.

[Sahota et al., 1995] Michael K. Sahota, Alan K. Mackworth, Rod A. Barman, and Stewart J. Kingdon. Real-time control of soccer-playing robots using off-board vision: the dynamite testbed. In *IEEE International Conference on Systems, Man, and Cybernetics*, pages 3690–3663, 1995.

[Sahota, 1993] Michael K. Sahota. Real-time intelligent behaviour in dynamic environments: Soccer-playing robots. Master's thesis, University of British Columbia, August 1993.

[Salganicoff, 1993] Marcos Salganicoff. Density-adaptive learning and forgetting. In *Proceedings of the Tenth International Conference on Machine Learning*, pages 276–283, 1993.

[Schlimmer and Granger, 1986] J.C. Schlimmer and R.H. Granger. Beyond incremental processing: Tracking concept drift. In *Proceedings of the Fiffth National Conference on Artificial Intelligence*, pages 502–507. Morgan Kaufman, Philadelphia, PA, 1986.

[Stone and Veloso, 1995a] Peter Stone and Manuela Veloso. Beating a defender in robotic soccer: Memory-based learning of a continuous function. Technical Report CMU-CS-95-222, Computer Science Department, Carnegie Mellon University, 1995.

[Stone and Veloso, 1995b] Peter Stone and Manuela Veloso. Broad learning from narrow training: A case study in robotic soccer. Technical Report CMU-CS-95-207, Computer Science Department, Carnegie Mellon University, 1995.

[Sutton and Whitehead, 1993] Richard S. Sutton and Steven D. Whitehead. Online learning with random representations. In *Proceedings of the Tenth International Conference on Machine Learning*, pages 314–321, 1993.

[Wettschereck and Dietterich, 1994] Dietrich Wettschereck and Thomas Dietterich. Locally adaptive nearest neighbor algorithms. In J. D. Cowan, G. Tesauro, and J. Alspector, editors, *Advances in Neural Information Processing Systems 6*, pages 184–191, San Mateo, CA, 1994. Morgan Kaufmann.

[Winstead and Christiansen, 1994] Nathaniel S. Winstead and Alan D. Christiansen. Pinball: Planning and learning in a dynamic real-time environment. In *AAAI-94 Fall Symposium on Control of the Physical World by Intelligent Agents*, pages 153–157, New Orleans, LA, November 1994.